# Improving Performance in Neural Networks Using a Boosting Algorithm

**Harris Drucker**
AT&T Bell Laboratories
Holmdel, NJ 07733

**Robert Schapire**
AT&T Bell Laboratories
Murray Hill, NJ 07974

**Patrice Simard**
AT&T Bell Laboratories
Holmdel, NJ 07733

## Abstract

A boosting algorithm converts a learning machine with error rate less than 50% to one with an arbitrarily low error rate. However, the algorithm discussed here depends on having a large supply of independent training samples. We show how to circumvent this problem and generate an ensemble of learning machines whose performance in optical character recognition problems is dramatically improved over that of a single network. We report the effect of boosting on four databases (all handwritten) consisting of 12,000 digits from segmented ZIP codes from the United State Postal Service (USPS) and the following from the National Institute of Standards and Testing (NIST): 220,000 digits, 45,000 upper case alphas, and 45,000 lower case alphas. We use two performance measures: the raw error rate (no rejects) and the reject rate required to achieve a 1% error rate on the patterns not rejected. Boosting improved performance in some cases by a factor of three.

## 1 INTRODUCTION

In this article we summarize a study on the effects of a boosting algorithm on the performance of an ensemble of neural networks used in optical character recognition problems. Full details can be obtained elsewhere (Drucker, Schapire, and Simard, 1993). The "boosting by filtering" algorithm is based on Schapire's original work (1990) which showed that it is theoretically possible to convert a learning machine with error rate less than 50% into an ensemble of learning machines whose error rate is arbitrarily low. The work detailed here is the first practical implementation of this boosting algorithm.

As applied to an ensemble of neural networks using supervised learning, the algorithm proceeds as follows: Assume an oracle that generates a large number of independent

training examples. First, generate a set of training examples and train a first network. After the first network is trained it may be used in combination with the oracle to produce a second training set in the following manner: Flip a fair coin. If the coin is heads, pass outputs from the oracle through the first learning machine until the first network misclassifies a pattern and add this pattern to a second training set. Otherwise, if the coin is tails pass outputs from the oracle through the first learning machine until the first network finds a pattern that it classifies correctly and add to the training set. This process is repeated until enough patterns have been collected. These patterns, half of which the first machine classifies correctly and half incorrectly, constitute the training set for the second network. The second network may then be trained.

The first two networks may then be used to produce a third training set in the following manner: Pass the outputs from the oracle through the first two networks. If the networks disagree on the classification, add this pattern to the training set. Otherwise, toss out the pattern. Continue this until enough patterns are generated to form the third training set. This third network is then trained.

In the final testing phase (of Schapire's original scheme), the test patterns (never previously used for training or validation) are passed through the three networks and labels assigned using the following voting scheme: If the first two networks agree, that is the label. Otherwise, assign the label as classified by the third network. However, we have found that if we add together the three sets of outputs from each of the three networks to obtain one set of ten outputs (for the digits) or one set of twenty-size outputs (for the alphas) we obtain better results. Typically, the error rate is reduced by .5% over straight voting.

The rationale for the better performance using addition is as follows: A voting criterion is a hard-decision rule. Each voter in the ensemble has an equal vote whether in fact the voter has high confidence (large difference between the two largest outputs in a particular network) or low confidence (small difference between the two largest outputs). By summing the outputs (a soft-decision rule) we incorporate the confidence of the networks into the total output. As will be seen later, this also allows us to build an ensemble with only two voters rather than three as called for in the original algorithm.

Conceptually, this process could be iterated in a recursive manner to produce an ensemble of nine networks, twenty-seven networks, etc. However, we have found significant improvement in going from one network to only three. The penalty paid is potentially an increase by a factor of three in evaluating the performance (we attribute no penalty to the increased training time). However it can show how to reduce this to a factor of 1.75 using sieving procedures.

## 2  A DEFORMATION MODEL

The proof that boosting works depends on the assumption of three independent training sets. Without a very large training set, this is not possible unless that error rates are large. After training the first network, unless the network has very poor performance, there are not enough remaining samples to generate the second training set. For example, suppose we had 9000 total examples and used the first 3000 to train the first network and that network achieves a 5% error rate. We would like the next training set to consist of 1500 patterns that the first network classifies incorrectly and 1500 that the first network

classifies incorrectly. At a 5% error rate, we need approximately 30,000 new images to pass through the first network to find 1500 patterns that the first network classifies incorrectly. These many patterns are not available. Instead we will generate additional patterns by using small deformations around the finite training set based on the techniques of Simard (Simard, et. al., 1992).

The image consists of a square pixel array (we use both 16x16 and 20x20). Let the intensity of the image at coordinate location (i,j) be $F_{ij}(x,y)$ where the (x,y) denotes that F is a differentiable and hence continuous function of x and y. i and j take on the discrete values 0,1,...,15 for a 16x16 pixel array.

The change in F at location (i,j) due to small x-translation, y-translation, rotation, diagonal deformation, axis deformation, scaling and thickness deformation is given by the following respective matrix inner products:

$$\Delta F_{ij}(x,y) = \left[ \frac{\partial F_{ij}(x,y)}{\partial x} \quad \frac{\partial F_{ij}(x,y)}{\partial y} \right]$$

$$x \left\{ k_1 \begin{bmatrix} 1 \\ 0 \end{bmatrix} + k_2 \begin{bmatrix} 0 \\ 1 \end{bmatrix} + k_3 \begin{bmatrix} -y \\ x \end{bmatrix} + k_4 \begin{bmatrix} y \\ x \end{bmatrix} + k_5 \begin{bmatrix} -x \\ y \end{bmatrix} + k_6 \begin{bmatrix} x \\ y \end{bmatrix} + k_7 \begin{bmatrix} \frac{\partial F_{ij}(x,y)}{\partial x} \\ \frac{\partial F_{ij}(x,y)}{\partial y} \end{bmatrix} \right\}$$

where the k's are small values and x and y are referenced to the center of the image. This construction depends on obtaining the two partial derivatives.

For example, if all the k's except $k_1$ are zero, then $\Delta F_{ij}(x,y) = k_1 \frac{\partial F_{ij}(x,y)}{\partial x}$ is the amount by which $F_{ij}(x,y)$ at coordinate location (i,j) changes due to an x-translation of value $k_1$.

The diagonal deformation can be conceived of as pulling on two opposite corners of the image thereby stretching the image along the 45 degree axis (away from the center) while simultaneously shrinking the image towards the center along a - 45 degree axis. If $k_4$ changes sign, we push towards the center along the 45 degree axis and pull away along the - 45 degree axis. Axis deformation can be conceived as pulling (or pushing) away from the center along the x-axis while pushing (or pulling) towards the center along the y-axis.

If all the k's except $k_7$ are zero, then $\Delta F_{ij}(x,y) = k_7 ||\nabla F_{ij}(x,y)||^2$ is the norm squared of the gradient of the intensity. It can be shown that this corresponds to varying the "thickness" of the image.

Typically the original image is very coarsely quantized and not differentiable. Smoothing of the original image is done by numerically convolving the original image with a 5x5 square kernel whose elements are values from the Gaussian: $\exp \frac{-(x^2+y^2)}{\sigma^2}$ to give us

a 16x16 or 20x20 square matrix of smoothed values.

A matrix of partial derivatives (with respect to x) for each pixel location is obtained by convolving the original image with a kernel whose elements are the derivatives with respect to x of the Gaussian function. We can similarly form a matrix of partial derivatives with respect to y. A new image can then be constructed by adding together the smoothed image and a differential matrix whose elements are given by the above equation.

Using the above equation, we may simulate an oracle by cycling through a finite sized training set, picking random values (uniformly distributed in some small range) of the constants k for each new image. The choice of the range of k is somewhat critical: too small and the new image is too close to the old image for the neural network to consider it a "new" pattern. Too large and the image is distorted and nonrepresentative of "real" data. We will discuss the proper choice of k later.

## 3 NETWORK ARCHITECTURES

We use as the basic learning machine a neural network with extensive use of shared weights (LeCun, et. al., 1989, 1990). Typically the number of weights is much less than the number of connections. We believe this leads to a better ability to reject images (i.e., no decision made) and thereby minimizes the number of rejects needed to obtain a given error rate on images not rejected. However, there is conflicting evidence (Martin & Pitman, 1991) that given enough training patterns, fully connected networks give similar performance to networks using weight sharing. For the digits there is a 16 by 16 input surrounded by a six pixel border to give a 28 by 28 input layer. The network has 4645 neurons, 2578 different weights, and 98442 connections.

The networks used for the alpha characters use a 20 by 20 input surrounded by a six pixel border to give a 32 by 32 input layer. There are larger feature maps and more layers, but essentially the same construction as for the digits.

## 4 TRAINING ALGORITHM

The training algorithm is described in general terms: Ideally, the data set should be broken up into a training set, a validation set and a test set. The training set and validation set are smoothed (no deformations) and the first network trained using a quasi-Newton procedure. We alternately train on the training data and test on the validation data until the error rate on the validation data reaches a minimum. Typically, there is some overtraining in that the error rate on the training data continues to decrease after the error rate on the validation set reaches a minimum.

Once the first network is trained, the second set of training data is generated by cycling deformed training data through the first network. After the pseudo-random tossing of a fair coin, if the coin is heads, deformed images are passed though the first network until the network makes a mistake. If tails, deformed images are passed through the network until the network makes a correct labeling. Each deformed image is generated from the original image by randomly selecting values of the constants k. It may require multiple passes through the training data to generate enough deformed images to form the second training set.

Recall that the second training set will consist equally of images that the first network misclassifies and images that the the first network classifies correctly. The total size of the training set is that of the first training set. Correctly classified images are not hard to find if the error rate of the first network is low. However, we only accept these images with probability 50%. The choice of the range of the random variables k should be such that the deformed images do not look distorted. The choice of the range of the k's is good if the error rate using the first network on the deformed patterns is approximately the same as the error rate of the first network on the validation set (NOT the first training set).

A second network is now trained on this new training set in the alternate train/test procedure using the original validation set (not deformed) as the test set. Since this training data is much more difficult to learn than the first training data, typically the error rate on the second training set using the second trained network will be higher (sometimes much higher) than the error rates of the first network on either the first training set or the validation set. Also, the error rate on the validation set using the second network will be higher than that of the first network because the network is trying to generalize from difficult training data, 50% of which the first network could not recognize.

The third training set is formed by once again generating deformed images and presenting the images to both the first and second networks. If the networks disagree (whether both are wrong or just one is), then that image is added to the third training set. The network is trained using this new training data and tested on the original validation set. Typically, the error rate on the validation set using the third network will be much higher than either of the first two networks on the same validation set.

The three networks are then tested on the third set of data, which is the smoothed test data. According to the original algorithm we should observe the outputs of the first two networks. If the networks agree, accept that labeling, otherwise use the labeling assigned by the third network. However, we are interested in more than a low error rate. We have a second criterion, namely the percent of the patterns we have to reject (i.e. no classification decision) in order to achieve a 1% error rate. The rationale for this is that if an image recognizer is used to sort ZIP codes (or financial statements) it is much less expensive to hand sort some numbers than to accept all and send mail to the wrong address or credit the wrong account. From now on we shall call this latter criterion the reject rate (without appending each time the statement "for a 1% error rate on the patterns not rejected").

For a single neural network, a reject criterion is to compare the two (of the ten or twenty-six) largest outputs of the network. If the difference is great, there is high confidence that the maximum output is the correct classification. Therefore, a critical threshold is set such that if the difference is smaller then that threshold, the image is rejected. The threshold is set so that the error rate on the patterns not rejected is 1%.

## 5  RESULTS

The boosting algorithm was first used on a database consisting of segmented ZIP codes from the United States Postal Service (USPS) divided into 9709 training examples and 2007 validation samples.

The samples supplied to us from the USPS were machine segmented from zip codes and labeled but not size normalized. The validation set consists of approximately 2% badly segmented characters (incomplete segmentations, decapitated fives, etc.) The training set was cleaned thus the validation set is significantly more difficult than the training set.

The data was size normalized to fit inside a 16x16 array, centered, and deslanted. There is no third group of data called the "test set" in the sense described previously even though the validation error rate has been commonly called the test error rate in prior work (LeCun, et. al., 1989, 1990).

Within the 9709 training digits are some machine printed digits which have been found to improve performance on the validation set. This data set has an interesting history having been around for three years with an approximate 5% error rate and 10% reject rate using our best neural network. There has been a slight improvement using double backpropagation (Drucker & LeCun, 1991) bringing down the error rate to 4.7% and the reject rate to 8.9% but nothing dramatic. This network, which has a 4.7% error rate was retrained on smoothed data by starting from the best set of weights. The second and third networks were trained as described previously with the following key numbers:

The retrained first network has a training error rate of less than 1%, a test error rate of 4.9% and a test reject rate of 11.5%

We had to pass 153,000 deformed images (recycling the 9709 training set) through the trained first network to obtain another 9709 training images. Of these 9709 images, approximately one-half are patterns that the first network misclassifies. This means that the first network has a 3.2% error rate on the deformed images, far above the error rate on the original training images.

A second network is trained and gives a 5.8% test error rate.

To generate the last training set we passed 195,000 patterns (again recycling the 9709) to give another set of 9709 training patterns. Therefore, the first two nets disagreed on 5% of the deformed patterns.

The third network is trained and gives a test error rate of 16.9%

Using the original voting scheme for these three networks, we obtained a 4.0% error rate, a significant improvement over the 4.9% using one network. As suggested before, adding together the three outputs gives a method of rejecting images with low confidence scores (when the two highest outputs are too close). For curiosity, we also determined what would happen if we just added together the first two networks:

Original network:    4.9% test error rate and 11.5% reject rate.
Two networks added:  3.9% test error rate and 7.9% reject rate.
Three networks added: 3.6% test error rate and  6.6% reject rate.

The ensemble of three networks gives a significant improvement, especially in the reject rate.

In April of 1992, the National Institute of Standards and Technology (NIST) provided a labeled database of 220,000 digits, 45,000 lower case alphas and 45,000 upper case

alphas. We divided these into training set, validation set, and test set. All data were resampled and size-normalized to fit into a 16x16 or 20x20 pixel array. For the digits, we deslanted and smoothed the data before retraining the first 16x16 input neural network used for the USPS data. After the second training set was generated and the second network trained the results from adding the two networks together were so good (Table 1) that we decided not to generate the third training set. For the NIST data, the error rates reported are those of the test data.

TABLE 1. Test error rate and reject rate in percent

| DATABASE | USPS digits | NIST digits | NIST upper alphas | NIST lower alpha |
|---|---|---|---|---|
| ERROR RATE SINGLE NET | 5.0 | 1.4 | 4.0 | 9.8 |
| ERROR RATE USING BOOSTING | 3.6 | .8 | 2.4 | 8.1 |
| REJECT RATE SINGLE NET | 9.6 | 1.0 | 9.2 | 29. |
| REJECT RATE USING BOOSTING | 6.6 | * | 3.1 | 21. |

* Reject rate is not reported if the error rate is below 1%.

# 6 CONCLUSIONS

In all cases we have been able to boost performance above that of single net. Although others have used ensembles to improve performance (Srihari, 1990; Benediktsson and Swain, 1992; Xu, et. al., 1992) the technique used here is particularly straightforward since the usual multi-classifier system requires a laborious development of each classifier. There is also a difference in emphasis. In the usual multi-classifier design, each classifier is trained independently and the problem is how to best combine the classifiers. In boosting, each network (after the first) has parameters that depend on the prior networks and we know how to combine the networks (by voting or adding).

# 7 ACKNOWLEDGEMENTS

We hereby acknowledge the United State Postal Service and the National Institute of Standards and Technology in supplying the databases.

## References

J.A. Benediktsson and P.H. Swain, "Consensus Theoretic Classification Methods", *IEEE trans. on Systems, Man, and Cybernetics*, Vol. 22, No. 4, July/August 1992, pp. 688-704.

H. Drucker, R. Schapire, and P. Simard "Boosting Performance in Neural Networks", *International Journal of Pattern Recognition and Artificial Intelligence*, (to be published, 1993)d

H. Drucker and Y. LeCun, "Improving Generalization Performance in Character Recognition", *Proceedings of the 1991 IEEE Workshop on Neural Networks for Signal Processing*, IEEE Press,pp. 198 - 207.

Y. LeCun, et. al., "Backpropagation Applied to Handwritten Zip Code Recognition", *Neural Computation 1*, 1989, pp. 541-551

Y. LeCun, et. al., Handwritten Digit Recognition with a Back-Propagation Network", In D.S. Touretsky (ed), *Advances in Neural Information Processing Systems 2*, (1990) pp. 396-404, San Mateo, CA: Morgan Kaufmann Publishers

G. L. Martin and J. A. Pitman, "Recognizing Handed-Printed Letters and Digits Using Backpropagation Learning", *Neural Computation*, Vol. 3, 1991, pp. 258-267.

R. Schapire, "The Strength of Weak Learnability", *Machine Learning*, Vol. 5, #2, 1990, pp. 197-227.

P. Simard, "Tangent Prop - A formalism for specifying selected invariances in an adaptive network", In J.E. Moody, S.J. Hanson, and R.P. Lippmann (eds.) *Advances in Neural Information Processing Systems 4*, (1992) p. 895-903, San Mateo, CA: Morgan Kaufmann Publishers

Sargur Srihari, "High-Performance Reading Machines", *Proceeding of the IEEE*, Vol 80, No. 7, July 1992, pp. 1120-1132.

C.Y. Suen, et. al., "Computer Recognition of Unconstrained Handwritten Numerals", *Proceeding of the IEEE*, Vol 80, No. 7, July 1992, pp. 1162-1180.

L. Xu, et. al., "Methods of Combining Multiple Classifiers", *IEEE Trans. on Systems Man, and Cybernetics*, Vol. 22, No. 3, May/June 1992, pp. 418-435.
